# Learning to Detect Natural Image Boundaries Using Brightness and Texture

**David R. Martin    Charless C. Fowlkes    Jitendra Malik**
Computer Science Division, EECS, U.C. Berkeley, Berkeley, CA 94720
{*dmartin,fowlkes,malik*} *@cs.berkeley.edu*

## Abstract

The goal of this work is to accurately detect and localize boundaries in natural scenes using local image measurements. We formulate features that respond to characteristic changes in brightness and texture associated with natural boundaries. In order to combine the information from these features in an optimal way, a classifier is trained using human labeled images as ground truth. We present precision-recall curves showing that the resulting detector outperforms existing approaches.

## 1   Introduction

Consider the image patches in Figure 1. Though they lack global context, it is clear which contain boundaries and which do not. The goal of this paper is to use features extracted from the image patch to estimate the posterior probability of a boundary passing through the center point. Such a local boundary model is integral to higher-level segmentation algorithms, whether based on grouping pixels into regions [21, 8] or grouping edge fragments into contours [22, 16].

The traditional approach to this problem is to look for discontinuities in image brightness. For example, the widely employed Canny detector [2] models boundaries as brightness step edges. The image patches show that this is an inadequate model for boundaries in natural images, due to the ubiquitous phenomenon of texture. The Canny detector will fire wildly inside textured regions where high-contrast contours are present but no boundary exists. In addition, it is unable to detect the boundary between textured regions when there is only a subtle change in average image brightness.

These significant problems have lead researchers to develop boundary detectors that explicitly model texture. While these work well on synthetic Brodatz mosaics, they have problems in the vicinity of brightness edges. Texture descriptors over local windows that straddle a boundary have different statistics from windows contained in either of the neighboring regions. This results in thin halo-like regions being detected around contours.

Clearly, boundaries in natural images are marked by changes in both texture and brightness. Evidence from psychophysics [18] suggests that humans make combined use of these two cues to improve detection and localization of boundaries. There has been limited work in computational vision on addressing the difficult problem of cue combination. For example, the authors of [8] associate a measure of texturedness with each point in an image in order to suppress contour processing in textured regions and vice versa. However, their solution is full of ad-hoc design decisions and hand chosen parameters.

The main contribution of this paper is to provide a more principled approach to cue combination by framing the task as a supervised learning problem. A large dataset of natural images that have been manually segmented by multiple human subjects [10] provides the ground truth label for each pixel as being on- or off-boundary. The task is then to model the probability of a pixel being on-boundary conditioned on some set of locally measured image features. This sort of quantitative approach to learning and evaluating boundary detectors is similar to the work of Konishi et al. [7] using the Sowerby dataset of English countryside scenes. Our work is distinguished by an explicit treatment of texture and brightness, enabling superior performance on a more diverse collection of natural images.

The outline of the paper is as follows. In Section 2 we describe the oriented energy and texture gradient features used as input to our algorithm. Section 3 discusses the classifiers we use to combine the local features. Section 4 presents our evaluation methodology along with a quantitative comparison of our method to existing boundary detection methods. We conclude in Section 5.

## 2  Image Features

### 2.1  Oriented Energy

In natural images, brightness edges are more than simple steps. Phenomena such as specularities, mutual illumination, and shading result in composite intensity profiles consisting of steps, peaks, and roofs. The oriented energy (OE) approach [12] can be used to detect and localize these composite edges [14]. OE is defined as:

$$\mathrm{OE}_{\theta,s} = (I * f_{\theta,s}^e)^2 + (I * f_{\theta,s}^o)^2$$

where $f_{\theta,s}^e$ and $f_{\theta,s}^o$ are a quadrature pair of even- and odd-symmetric filters at orientation $\theta$ and scale $s$. Our even-symmetric filter is a Gaussian second-derivative, and the corresponding odd-symmetric filter is its Hilbert transform. $\mathrm{OE}_{\theta,s}$ has maximum response for contours at orientation $\theta$. We compute OE at 3 half-octave scales starting at $s = 0.7\%$ the image diagonal. The filters are elongated by a ratio of 3:1 along the putative boundary direction.

### 2.2  Texture Gradient

We would like a directional operator that measures the degree to which texture varies at a location $(x, y)$ in direction $\theta$. A natural way to operationalize this is to consider a disk of radius $s$ centered on $(x, y)$, and divided in two along a diameter at orientation $\theta$. We can then compare the texture in the two half discs with some texture dissimilarity measure. Oriented texture processing along these lines has been pursued by [19].

What texture dissimilarity measure should one use? There is an emerging consensus that for texture analysis, an image should first be convolved with a bank of filters tuned to various orientations and spatial frequencies [4, 9]. After filtering, a texture descriptor is then constructed using the empirical distribution of filter responses in the neighborhood of a pixel. This approach has been shown to be very powerful both for texture synthesis [5] as well as texture discrimination [15].

Puzicha et al. [15] evaluate a wide range of texture descriptors in this framework. We choose the approach developed in [8]. Convolution with a filter bank containing both even and odd filters at multiple orientations as well as a radially symmetric center-surround filter associates a vector of filter responses to every pixel. These vectors are clustered using k-means and each pixel is assigned to one of the cluster centers, or *textons*. Texture dissimilarities can then be computed by comparing the histograms of textons in the two disc halves. Let $g_i$ and $h_i$ count how many pixels of texton type $i$ occur in each half disk.

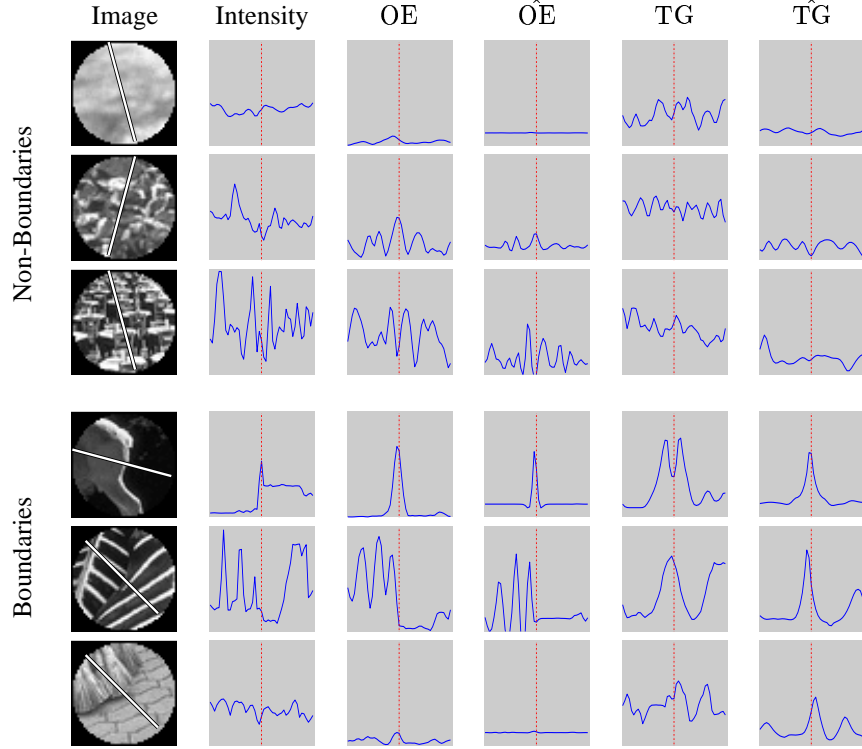

Figure 1: Local image features. In each row, the first panel shows the image patch. The following panels show feature profiles along the line marked in each patch. The features are raw image intensity, raw oriented energy OE, localized oriented energy ÔE, raw texture gradient TG, and localized texture gradient T̂G. The vertical line in each profile marks the patch center. The challenge is to combine these features in order to detect and localize boundaries.

We define the texture gradient (TG) to be the $\chi^2$ distance between these two histograms:

$$\chi^2(g, h) = \frac{1}{2} \sum \frac{(g_i - h_i)^2}{g_i + h_i}$$

The texture gradient is computed at each pixel $(x, y)$ over 12 orientations and 3 half-octave scales starting at $s = 1.5\%$ of the image diagonal.

## 2.3 Localization

The underlying function we are trying to learn is tightly peaked around the location of image boundaries marked by humans. In contrast, Figure 1 shows that the features we have discussed so far don't have this structure. By nature of the fact that they pool information over some support, they produce smooth, spatially extended outputs. The texture gradient is particularly prone to this effect, since the texture in a window straddling the boundary is distinctly different than the textures on either side of the boundary. This often results in a wide plateau or even double peaks in the texture gradient.

Since each pixel is classified independently, these spatially extended features are particularly problematic as both on-boundary pixels and nearby off-boundary pixels will have large OE and TG. In order to make this spatial structure available to the classifier we transform the raw OE and TG signals in order to emphasize local maxima. Given a feature $f(x)$

defined over spatial coordinate $x$ orthogonal to the edge orientation, consider the derived feature $\hat{f}(x) = f(x)/d(x)$, where $d(x) = -|f'(x)|/f''(x)$ is the first-order approximation of the distance to the nearest maximum of $f(x)$. We use the stabilized version

$$\hat{f}(x) = f(x) \cdot \left( \frac{-f''(x)}{|f'(x)| + \epsilon} \right) \tag{1}$$

with $\epsilon$ chosen to optimize the performance of the feature. By incorporating the $1/d(x)$ localization term, $\hat{f}(x)$ will have narrower peaks than the raw $f(x)$.

To robustly estimate the directional derivatives and localize the peaks, we fit a cylindrical parabola over a circular window of radius $r$ centered at each pixel. The coefficients of the quadratic fit $ax^2 + bx + c$ provide directly the signal derivatives, so the transform above becomes $\hat{f} = -(2c^+a^+)/(|b| + \epsilon)$, where $c$ and $a$ require half-wave rectification.[1] This transformation is applied to the oriented energy and texture gradient signals at each orientation $\theta$ and scale $s$ separately. In order to set $r$ and $\epsilon$, we optimized the performance of each feature independently with respect to the training data.[2]

Columns 4 and 6 in Figure 1 show the results of applying this transformation which clearly has the effect of reducing noise and tightly localizing the boundaries. Our final feature set consists of these localized signals $\hat{TG}$ and $\hat{OE}$, each at three scales. This yields a 6-element feature vector at 12 orientations at each pixel.

## 3   Cue Combination Using Classifiers

We would like to combine the cues given by the local feature vector in order to estimate the posterior probability of a boundary at each image location $(x, y, \theta)$. Previous work on learning boundary models includes [11, 7]. We consider several parametric and non-parametric models, covering a range of complexity and computational cost. The simplest are able to capture the complementary information in the 6 features. The more powerful classifiers have the potential to capture non-linear cue "gating" effects. For example, one may wish to ignore brightness edges inside high-contrast textures where OE is high and TG is low. These are the classifiers we use:

***Density Estimation*** Adaptive bins are provided by vector quantization using k-means. Each centroid provides the density estimate of its Voronoi cell as the fraction of on-boundary samples in the cell. We use k=128 and average the estimates from 10 runs.

***Classification Trees*** The domain is partitioned hierarchically. Top-down axis-parallel splits are made so as to maximize the information gain. A 5% bound on the error of the density estimate is enforced by splitting cells only when both classes have >400 points present.

***Logistic Regression*** This is the simplest of our classifiers, and the one perhaps most easily replicated by neurons in the visual cortex. Initialization is random, and convergence is fast and reliable by maximizing the likelihood. We also consider two variants: quadratic combinations of features, and boosting using the confidence-rated generalization of AdaBoost by Schapire and Singer [20]. No more than 10 rounds of boosting are required for this problem.

***Hierarchical Mixtures of Experts*** The HME model of Jordan and Jacobs [6] is a mixture model where both the components and mixing coefficients are fit by logistic functions. We

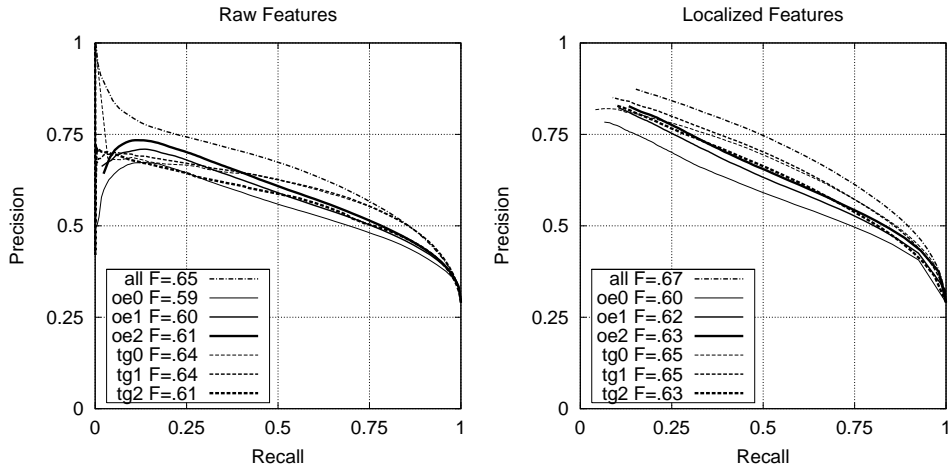

Figure 2: Performance of raw (left) and localized features (right). The precision and recall axes are described in Section 4. Curves towards the top (lower noise) and right (higher accuracy) are more desirable. Each curve is scored by the F-measure, the value of which is shown in the legend. In all the precision-recall graphs in this paper, the maximum F-measure occurs at a recall of approximately 75%. The left plot shows the performance of the raw OE and TG features using the logistic regression classifier. The right plot shows the performance of the features after applying the localization process of Equation 1. It is clear that the localization function greatly improves the quality of the individual features, especially the texture gradient. The top curve in each graph shows the performance of the features in combination. While tuning each feature's $\{\epsilon, r\}$ parameters individually is suboptimal, overall performance still improves.

consider small binary trees up to a depth of 3 (8 experts). The model is initialized in a greedy, top-down manner and fit with EM.

***Support Vector Machines***  We use the SVM package `libsvm` [3] to do soft-margin classification using Gaussian kernels. The optimal parameters were $\nu$=0.2 and $\sigma$=0.2.

The ground truth boundary data is based on the dataset of [10] which provides 5-6 human segmentations for each of 1000 natural images from the Corel image database. We used 200 images for training and algorithm development. The 100 test images were used only to generate the final results for this paper. The authors of [10] show that the segmentations of a single image by the different subjects are highly consistent, so we consider all human-marked boundaries valid. We declare an image location $(x, y, \theta)$ to be on-boundary if it is within $\Delta x$=2 pixels and $\Delta \theta$=30 degrees of any human-marked boundary. The remainder are labeled off-boundary.

This classification task is characterized by relatively low dimension, a large amount of data (100M samples for our 240x160-pixel images), and poor separability. The maximum feasible amount of data, uniformly sampled, is given to each classifier. This varies from 50M samples for density estimation to 20K samples for the SVM. Note that a high degree of class overlap in *any* local feature space is inevitable because the human subjects make use of both global constraints and high-level information to resolve locally ambiguous boundaries.

## 4   Results

The output of each classifier is a set of oriented $P_b$ *images*, which provide the probability of a boundary at each image location $(x, y, \theta)$ based on local information. For several of the

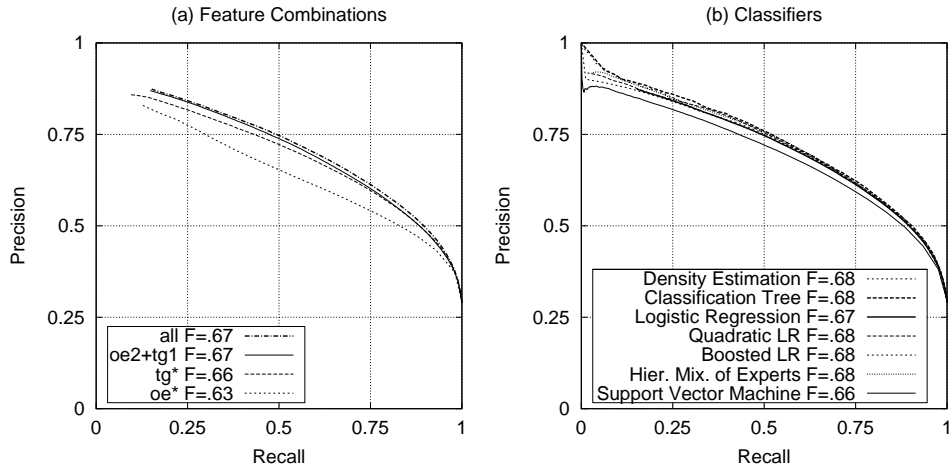

Figure 3: Precision-recall curves for (a) different feature combinations, and (b) different classifiers. The left panel shows the performance of different combinations of the localized features using the logistic regression classifier: the 3 OE features (oe*), the 3 TG features (tg*), the best performing single OE and TG features (oe2+tg1), and all 6 features together. There is clearly independent information in each feature, but most of the information is captured by the combination of one OE and one TG feature. The right panel shows the performance of different classifiers using all 6 features. All the classifiers achieve similar performance, except for the SVM which suffers from the poor separation of the data. Classification trees performs the best by a slim margin. Based on performance, simplicity, and low computation cost, we favor the logistic regression and its variants.

classifiers we consider, the $P_b$ image provides actual posterior probabilities, which is particularly appropriate for the local measurement model in higher-level vision applications. For the purpose of evaluation, we take the maximum $P_b$ over orientations.

In order to evaluate the boundary model against the human ground truth, we use the *precision-recall* framework, a standard evaluation technique in the information retrieval community [17]. It is closely related to the ROC curves used for by [1] to evaluate boundary models. The precision-recall curve captures the trade-off between accuracy and noise as the detector threshold is varied. Precision is the fraction of detections which are true positives, while recall is the fraction of positives that are detected. These are computed using a distance tolerance of 2 pixels to allow for small localization errors in both the machine and human boundary maps.

The precision-recall curve is particularly meaningful in the context of boundary detection when we consider applications that make use of boundary maps, such as stereo or object recognition. It is reasonable to characterize higher level processing in terms of how much true signal is required to succeed, and how much noise can be tolerated. Recall provides the former and precision the latter. A particular application will define a relative cost $\alpha$ between these quantities, which focuses attention at a specific point on the precision-recall curve. The *F-measure*, defined as $F = PR/(\alpha R + (1 - \alpha)P)$, captures this trade-off. The location of the maximum F-measure along the curve provides the optimal threshold given $\alpha$, which we set to 0.5 in our experiments.

Figure 2 shows the performance of the raw and localized features. This provides a clear quantitative justification for the localization process described in Section 2.3. Figure 3a shows the performance of various linear combinations of the localized features. The combination of multiple scales improves performance, but the largest gain comes from using OE and TG together.

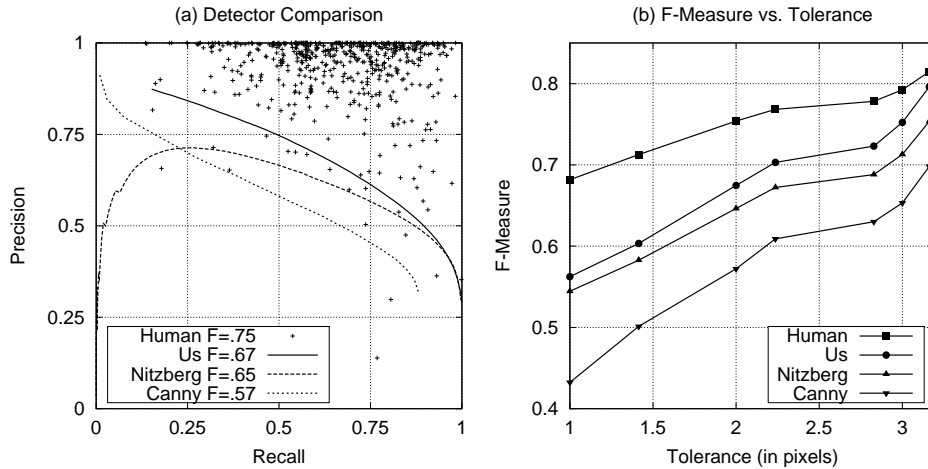

Figure 4: The left panel shows precision-recall curves for a variety of boundary detection schemes, along with the precision and recall of the human segmentations when compared with each other. The right panel shows the F-measure of each detector as the distance tolerance for measuring precision and recall varies. We take the Canny detector as the baseline due to its widespread use. Our detector outperforms the learning-based Nitzberg detector proposed by Konishi et al. [7], but there is still a significant gap with respect to human performance.

The results presented so far use the logistic regression classifier. Figure 3b shows the performance of the 7 different classifiers on the complete feature set. The most obvious trend is that they all perform similarly. The simple non-parametric models – the classification tree and density estimation – perform the best, as they are most able to make use of the large quantity of training data to provide unbiased estimates of the posterior. The plain logistic regression model performs extremely well, with the variants of logistic regression – quadratic, boosted, and HME – performing only slightly better. The SVM is a disappointment because of its lower performance, high computational cost, and fragility. These problems result from the non-separability of the data, which requires >20% of the training examples to be used as support vectors. Balancing considerations of performance, model complexity, and computational cost, we favor the logistic model and its variants.[3]

Figure 4 shows the performance of our detector compared to two other approaches. Because of its widespread use, MATLAB's implementation of the classic Canny [2] detector forms the baseline. We also consider the Nitzberg detector [13, 7], since it is based on a similar supervised learning approach, and Konishi et al. [7] show that it outperforms previous methods. To make the comparisons fair, the parameters of both Canny and Nitzberg were optimized using the training data. For Canny, this amounts to choosing the optimal scale. The Nitzberg detector generates a feature vector containing eigenvalues of the 2nd moment matrix; we train a classifier on these 2 features using logistic regression.

Figure 4 also shows the performance of the human data as an upper-bound for the algorithms. The human precision-recall points are computed for each segmentation by comparing it to the other segmentations of the same image. The approach of this paper is a clear improvement over the state of the art in boundary detection, but it will take the addition of high-level and global information to close the gap between the machine and human performance.

# 5 Conclusion

We have defined a novel set of brightness and texture cues appropriate for constructing a local boundary model. By using a very large dataset of human-labeled boundaries in natural images, we have formulated the task of cue combination for local boundary detection as a supervised learning problem. This approach models the true posterior probability of a boundary at every image location and orientation, which is particularly useful for higher-level algorithms. Based on a quantitative evaluation on 100 natural images, our detector outperforms existing methods.

## Footnotes

[1] Windowed parabolic fitting is known as 2nd-order Savitsky-Golay filtering. We also considered Gaussian derivative filters $\{G_r, G'_r, G''_r\}$ to estimate $\{f_r, f'_r, f''_r\}$ with nearly identical results.

[2] The fitted values are $\epsilon=\{0.1, 0.075, 0.013\}$ and $r=\{2.1, 2.5, 3.1\}$ for OE, and $\epsilon=\{.057, .016, .005\}$ and $r=\{6.66, 9.31, 11.72\}$ for TG. $r$ is measured in pixels.

[3]The fitted coefficients for the logistic are {.088,-.029,.019} for OE and {.31,.26,.27} for TG, with an offset of -2.79. The features have been separately normalized to have unit variance.

## References

[1] K. Bowyer, C. Kranenburg, and S. Dougherty. Edge detector evaluation using empirical ROC curves. *Proc. IEEE Conf. Comput. Vision and Pattern Recognition*, 1999.

[2] J. Canny. A computational approach to edge detection. *IEEE Trans. Pattern Analysis and Machine Intelligence*, 8:679–698, 1986.

[3] C. Chang and C. Lin. *LIBSVM: a library for support vector machines*, 2001. Software available at http://www.csie.ntu.edu.tw/~cjlin/libsvm.

[4] I. Fogel and D. Sagi. Gabor filters as texture discriminator. *Bio. Cybernetics*, 61:103–13, 1989.

[5] D. J. Heeger and J. R. Bergen. Pyramid-based texture analysis/synthesis. In *Proceedings of SIGGRAPH '95*, pages 229–238, 1995.

[6] M. I. Jordan and R. A. Jacobs. Hierarchical mixtures of experts and the EM algorithm. *Neural Computation*, 6:181–214, 1994.

[7] S. Konishi, A. L. Yuille, J. Coughlan, and S. C. Zhu. Fundamental bounds on edge detection: an information theoretic evaluation of different edge cues. *Proc. IEEE Conf. Comput. Vision and Pattern Recognition*, pages 573–579, 1999.

[8] J. Malik, S. Belongie, T. Leung, and J. Shi. Contour and texture analysis for image segmentation. *Int'l. Journal of Computer Vision*, 43(1):7–27, June 2001.

[9] J. Malik and P. Perona. Preattentive texture discrimination with early vision mechanisms. *J. Optical Society of America*, 7(2):923–32, May 1990.

[10] D. Martin, C. Fowlkes, D. Tal, and J. Malik. A database of human segmented natural images and its application to evaluating segmentation algorithms and measuring ecological statistics. In *Proc. 8th Int'l. Conf. Computer Vision*, volume 2, pages 416–423, July 2001.

[11] M. Meilă and J. Shi. Learning segmentation by random walks. In *NIPS*, 2001.

[12] M.C. Morrone and D.C. Burr. Feature detection in human vision: a phase dependent energy model. *Proc. R. Soc. Lond. B*, 235:221–45, 1988.

[13] M. Nitzberg, D. Mumford, and T. Shiota. *Filtering, Segmentation and Depth*. Springer-Verlag, 1993.

[14] P. Perona and J. Malik. Detecting and localizing edges composed of steps, peaks and roofs. In *Proc. Int. Conf. Computer Vision*, pages 52–7, Osaka, Japan, Dec 1990.

[15] J. Puzicha, T. Hofmann, and J. Buhmann. Non-parametric similarity measures for unsupervised texture segmentation and image retrieval. In *Computer Vision and Pattern Recognition*, 1997.

[16] X. Ren and J. Malik. A probabilistic multi-scale model for contour completion based on image statistics. *Proc. 7th Europ. Conf. Comput. Vision*, 2002.

[17] C. Van Rijsbergen. *Information Retrieval, 2nd ed.* Dept. of Comp. Sci., Univ. of Glasgow, 1979.

[18] J. Rivest and P. Cavanagh. Localizing contours defined by more than one attribute. *Vision Research*, 36(1):53–66, 1996.

[19] Y. Rubner and C. Tomasi. Coalescing texture descriptors. *ARPA Image Understanding Workshop*, 1996.

[20] R. E. Schapire and Y. Singer. Improved boosting algorithms using confidence-rated predictions. *Machine Learning*, 37(3):297–336, 1999.

[21] Z. Tu, S. Zhu, and H. Shum. Image segmentation by data driven markov chain monte carlo. In *Proc. 8th Int'l. Conf. Computer Vision*, volume 2, pages 131–138, July 2001.

[22] L.R. Williams and D.W. Jacobs. Stochastic completion fields: a neural model of illusory contour shape and salience. In *Proc. 5th Int. Conf. Computer Vision*, pages 408–15, June 1995.
